# Feature Selection in Clustering Problems

**Volker Roth and Tilman Lange**
ETH Zurich, Institut f. Computational Science
Hirschengraben 84, CH-8092 Zurich
Tel: +41 1 6323179
{*vroth, tilman.lange*}*@inf.ethz.ch*

## Abstract

A novel approach to combining *clustering* and *feature selection* is presented. It implements a *wrapper* strategy for feature selection, in the sense that the features are directly selected by optimizing the discriminative power of the used partitioning algorithm. On the technical side, we present an efficient optimization algorithm with guaranteed local convergence property. The only free parameter of this method is selected by a resampling-based stability analysis. Experiments with real-world datasets demonstrate that our method is able to infer both meaningful partitions and meaningful subsets of features.

## 1  Introduction

The task of selecting relevant features in classification problems can be viewed as one of the most fundamental problems in the field of machine learning. A major motivation for selecting a subset of features from which a learning rule is constructed is the interest in *sparse and interpretable* rules, emphasizing only a few relevant variables. In *supervised* learning scenarios, feature selection has been studied widely in the literature. The methods used can be subdivided in *filter* methods and *wrapper* methods. The main difference is that a wrapper method makes use of the classifier, while a filter method does not. From a conceptual viewpoint, wrapper approaches are clearly advantageous, since the features are selected by optimizing the discriminative power of the finally used classifier.

Selecting features in *unsupervised* learning scenarios is a much harder problem, due to the absence of class labels that would guide the search for relevant information. Problems of this kind have been rarely studied in the literature, for exceptions see e.g. [1, 9, 15]. The common strategy of most approaches is the use of an iterated stepwise procedure: in the first step a set of hypothetical partitions is extracted (the *clustering* step), and in the second step features are scored for relevance (the *relevance determination* step). A possible shortcoming is the way of combining these two steps in an "ad hoc" manner: usually the relevance determination mechanism implements a filter approach and does not take into account the properties of the clustering method used. Usual scoring methods make an implicit independence assumption, while ignoring feature correlations. It is thus of particular interest to combine *wrapper* selection strategies and clustering methods. The approach presented in this paper can be viewed as a method of this kind. It combines a Gaussian mixture model with a Bayesian feature selection principle. The usual combinatorial problems involved with wrapper approaches are overcome by using a Bayesian marginalization mechanism. We present an efficient optimization algorithm for our model with guaranteed convergence to a local optimum.

The only free model parameter is selected by a resampling-based *stability analysis*. The problem of many ambiguous and equally high-scoring splitting hypotheses, which seems to

be a an inherent shortcoming of many other approaches, is successfully overcome. A comparison with ground-truth labels in control experiments indicates that the selected models induce sample clusters and feature subsets which both provide a clear interpretation.

Our approach to combining clustering and feature selection is based on a Gaussian mixture model, which is optimized by way of the classical *expectation-maximization* (EM) algorithm. In order to incorporate the feature selection mechanism, the M-step is first reformulated as a linear discriminant analysis (LDA) problem, which makes use of the "fuzzy labels" estimated in the preceding E-step. We then use the well-known identity of LDA and linear regression to restate the M-step in a form which easily allows us to regularize the estimation problem by specifying a prior distribution over the regression coefficients. This distribution has the functional form of an *Automatic Relevance Determination* (ARD) prior. For each regression coefficient, the ARD prior contains a free hyperparameter, which encodes the "relevance" of the corresponding variable in the linear regression. In a Bayesian marginalization step, these hyperparameters are then integrated out. We finally arrive at an M-step with integrated feature selection mechanism.

## 2  Clustering and Bayesian relevance determination

**Gaussian mixtures and LDA.** The dataset is given as a collection of $N$ samples $\boldsymbol{x}_i \in \mathbb{R}^d$. For the purpose of finding *clusters*, consider now a Gaussian mixture model with 2 mixture components which share an identical covariance matrix $\Sigma$. Under this model, the log-likelihood for the dataset reads

$$l^{mix} = \sum_{i=1}^{N} \log \left( \sum_{\nu=1}^{2} \pi_\nu \phi(\boldsymbol{x}_i; \boldsymbol{\mu}_\nu, \Sigma) \right), \tag{1}$$

where the mixing proportions $\pi_\nu$ sum to one, and $\phi$ denotes a Gaussian density. The classical EM-algorithm, [2], provides a convenient method for maximizing $l^{mix}$:

> **E-step:** set $p_{\eta i} = \mathsf{Prob}(\boldsymbol{x}_i \in \text{class } \eta) = \dfrac{\pi_\eta \phi(\boldsymbol{x}_i; \boldsymbol{\mu}_\eta, \Sigma)}{\sum_{\nu=1}^{2} \pi_\nu \phi(\boldsymbol{x}_i; \boldsymbol{\mu}_\nu, \Sigma)}$.
>
> **M-step:** set $\boldsymbol{\mu}_\nu = \dfrac{\sum_{i=1}^{N} p_{\nu i} \boldsymbol{x}_i}{\sum_{i=1}^{N} p_{\nu i}}$, $\quad \Sigma = \dfrac{1}{N} \sum_{\nu=1}^{2} \sum_{i=1}^{N} p_{\nu i} (\boldsymbol{x}_i - \boldsymbol{\mu}_\nu)(\boldsymbol{x}_i - \boldsymbol{\mu}_\nu)^\top$.

The likelihood equations in the M-step can be viewed as weighted mean and covariance maximum likelihood estimates in a weighted and augmented problem: one replicates the $N$ observations 2 times, with the $\nu$-th such replication having observation weights $p_{\nu i}$. In [5] it is proven that the M-step can be carried out via a weighted and augmented *linear discriminant analysis* (LDA). Following [6], any LDA problem can be restated as an *optimal scoring* problem. Let the class-memberships of the $N$ data vectors be coded as a matrix $Z$, the $i, \nu$-th entry of which equals one if the $i$-th observation belongs to class $\nu$. The point of optimal scoring is to turn categorical variables into quantitative ones: the score vector $\boldsymbol{\theta}$ assigns the real number $\theta_\nu$ to the entries in the $\nu$-th column of $Z$. The simultaneous estimation of scores and regression coefficients $\boldsymbol{\beta}$ constitutes the optimal scoring problem: minimize

$$M(\boldsymbol{\theta}, \boldsymbol{\beta}) = \|Z\boldsymbol{\theta} - X\boldsymbol{\beta}\|_2^2 \tag{2}$$

under the constraint $\frac{1}{N}\|Z\boldsymbol{\theta}\|_2^2 = 1$. The notion $\|\cdot\|_2^2$ stands for the squared $\ell_2$–norm, and $X$ denotes the (centered) data matrix of dimension $N \times d$. In [6] an algorithm for carrying out this optimization has been proposed, whose main ingredient is a linear regression of the data matrix $X$ against the scored indicator matrix $Z\boldsymbol{\theta}$.

Returning from a standard LDA-problem to the above weighted and augmented problem, it turns out that it is *not* necessary to explicitly replicate the observations: the optimal scoring version of LDA allows an implicit solution of the augmented problem that still uses only $N$ observations. Instead of using a response indicator matrix $Z$, a *blurred* response matrix $\tilde{Z}$ is employed, whose rows consist of the current class probabilities for each observation. At each M-step this $\tilde{Z}$ enters in the linear regression, see [5]. After iterated application of the E- and M-step, an observation $\boldsymbol{x}_i$ is finally assigned to the class $\nu$ with highest probability of membership $p_{\nu i}$. Note that the EM-iterations converge to a local maximum.

**LDA and Automatic Relevance Determination.** We now focus on incorporating the automatic feature selection mechanism into the EM-algorithm. According to [6], the 2-class LDA problem in the M-step can be solved by the following algorithm:

---
1. Choose an initial $N$-vector of scores $\boldsymbol{\theta}_0$ which satisfies $N^{-1}\boldsymbol{\theta}_0^T \tilde{Z}^T \tilde{Z} \boldsymbol{\theta}_0 = 1$, and is orthogonal to a $k$-vector of ones, $\mathbf{1}_k$. Set $\boldsymbol{\theta}^* = \tilde{Z}\boldsymbol{\theta}_0$;
2. Run a linear regression of $X$ on $\boldsymbol{\theta}^*$: $\widehat{\boldsymbol{\theta}^*} = X(X^T X)^{-1}X^T \boldsymbol{\theta}^* \equiv X\boldsymbol{\beta}$.

---

The feature selection mechanism can now be incorporated in the M-step by imposing a certain constraint on the linear regression. In [6, 4] it has been proposed to use a ridge-type penalized regression. Taking a Bayesian perspective, such a ridge-type penalty can be interpreted as introducing a spherical Gaussian prior over the coefficients: $p(\boldsymbol{\beta}) = \mathcal{N}(\mathbf{0}, \lambda^{-1}I)$. The main idea of incorporating an automatic feature selection mechanism consists of replacing the Gaussian prior with an *automatic relevance determination* (ARD) prior[1] of the form

$$p(\boldsymbol{\beta}|\,\boldsymbol{\vartheta}) = \prod_i \mathcal{N}(0, \vartheta_i^{-1}) \propto \exp[-\sum_i \vartheta_i \beta_i^2]. \tag{3}$$

In this case, each coefficient $\beta_i$ has its own prior variance $\vartheta_i^{-1}$. Note that in the above ARD framework only the functional form of the prior (3) is fixed, whereas the parameters $\vartheta_i$, which encode the "relevance" of each variable, are estimated from the data. In [3] the following Bayesian inference procedure for the prior parameters has been introduced: given exponential hyperpriors (the variances $\vartheta_i^{-1}$ must be nonnegative), $p(\vartheta_i) = \frac{\gamma}{2}\exp\{-\frac{\gamma \vartheta_i}{2}\}$, one can analytically integrate out the hyperparameters from the prior distribution over the coefficients $\beta_i$:

$$p(\beta_i) = \int_0^\infty p(\beta_i|\vartheta_i)p(\vartheta_i)\,d\vartheta_i = \frac{\gamma}{2}\exp\{-\sqrt{\gamma}|\beta_i|\}. \tag{4}$$

Switching to the maximum a posteriori (MAP) solution in log-space, this marginalization directly leads us to the following penalized functional:

$$M(\boldsymbol{\theta}, \boldsymbol{\beta}) = \|\tilde{Z}\boldsymbol{\theta} - X\boldsymbol{\beta}\|_2^2 + \tilde{\lambda}\,\|\boldsymbol{\beta}\|_1, \tag{5}$$

where $\tilde{\lambda} \equiv \sqrt{\gamma}$ has the role of a Lagrange parameter in the $\ell_1$–constrained problem: minimize $\|\tilde{Z}\boldsymbol{\theta} - X\boldsymbol{\beta}\|_2^2$ subject to $\|\boldsymbol{\beta}\|_1 < \kappa$. In the statistical literature, this model is known as the *Least Absolute Shrinkage and Selection Operator* (LASSO) model, [14].

Returning to equation (3), we are now able to interpret the LASSO estimate as a Bayesian feature selection principle: for the purpose of feature selection, we would like to estimate the value of a binary selection variable $\mathcal{S}$ for each feature: $\mathcal{S}_i$ equals one, if the $i$-th feature is considered relevant for the given task, and zero otherwise. Taking into account feature correlations, estimation of $\mathcal{S}_i$ necessarily involves searching the space of all possible subsets of features containing the $i$-th one. In the Bayesian ARD formalism, this combinatorial explosion of the search space is overcome by relaxing the binary selection variable to a positive real-valued variance of a Gaussian prior over each component of the coefficient vector. Following the Bayesian inference principle, we introduce hyperpriors and integrate out these variances, and we finally arrive at the $\ell_1$–constrained LASSO problem.

**Optimizing the final model.** Since space here precludes a detailed discussion of $\ell_1$–constrained regression problems, the reader is referred to [12], where a highly efficient algorithm with guaranteed global convergence has been proposed. Given this *global* convergence in the M-step, for the EM-model we can guarantee convergence to a *local* maximum of the constrained likelihood. Consider two cases: (i) the unconstrained solution is feasible. In this case our algorithm simply reduces to the standard EM procedure, for which is it known that in every iteration the likelihood monotonically increases; (ii) the $\ell_1$–constraint is active. Then, in every iteration the LASSO algorithm maximizes the likelihood within the feasible region of $\beta$-values defined by $\|\boldsymbol{\beta}\|_1 < \kappa$. The likelihood cannot be decreased in further stages of the iteration, since any solution $\boldsymbol{\beta}$ found in a preceding iteration is also a valid solution for the actual problem (note that $\kappa$ is fixed!). In this case, the algorithm has converged to a local maximum of the likelihood within the constraint region.

# 3 Model selection

Our model has only one free parameter, namely the value of the $\ell_1$–constraint $\kappa$. In the following we describe a method for selecting $\kappa$ by observing the *stability* of data partitions. For each of the partitions which we have identified as "stable", we then examine the fluctuations involved in the feature selection process. It should be noticed that the concept of measuring the stability of solutions as a means of model selection has been successfully applied to several unsupervised learning problems, see e.g. [8, 11].

We will usually find many potential splits of a dataset, depending on how many features are selected: if we select only one feature, it is likely to find many competing hypotheses for splits. The problem is that most of the feature vectors usually vote for a different partition. If, on the other hand, we select too many features, we face the usual problems of finding structure in high-dimensional datasets: our functional which we want to optimize will have many local minima, and with high probability, the EM-algorithm will find suboptimal solutions. Between these two extremes, we can hope to find relatively stable splits, which are robust against noise and also against inherent instabilities of the optimization method.

To obtain a quantitative measure of stability, we propose the following procedure: run the class discovery method once, corrupt the data vectors by a small amount of noise, repeat the grouping procedure, and calculate the Hamming distance between the two partitions as a measure of (in-)stability. For computing Hamming distances, the partitions are viewed as vectors containing the cluster labels. Simply taking the average stability over many such two-sample comparisons, however, would not allow an adequate handling of situations where there are two equally likely stable solutions, of which the clustering algorithm randomly selects one. In such situations, the averaged stability will be very low, despite the fact that there exist two stable splitting hypotheses. This problem can be overcome by looking for compact *clusters* of highly similar partitions, leading to the following algorithm:

---
**Algorithm for identifying stable partitions:** for different values of the $\ell_1$–constraint $\kappa$ do
(i) compute $m$ noisy replications of the data
(ii) run the class discovery algorithm for each of these datasets
(iii) compute the $m \times m$ matrix of pairwise Hamming distances between all partitions
(iv) cluster the partitions into compact groups and score the groups by their frequency
(v) select dominant groups of partitions and choose representative partitions

---

In step (i) a "suitable" noise level must be chosen a priori. In our experiments we make use of the fact that we have normalized the input data to have zero mean and unit variance. Given this normalization, we then add Gaussian noise with $5\%$ of the total variance in the dataset, i.e. $\sigma^2 = 0.05$. In step (iii) we use Hamming distances as a dissimilarity measure between partitions. In order to make Hamming distances suitable for this purpose, we have to consider the inherent permutation symmetry of the clustering process: a cluster called "1" in the first partition can be called "2" in the second one. When computing the pairwise Hamming distances, we thus have to minimize over the two possible permutations of cluster labels. Steps (iv) and (v) need some further explanation: the problem of identifying compact groups in datasets which are represented by pairwise distances can by solved by optimizing the *pairwise clustering cost function*, [7]. We iteratively increase the number of clusters (which is a free parameter in the pairwise clustering functional) until the average dissimilarity in each group does not exceed a predefined threshold. Reasonable problem-specific thresholds can be defined by considering the following null-model: given $N$ samples, the average Hamming distance between two randomly drawn 2–partitions $\mathcal{P}_1$ and $\mathcal{P}_2$ is roughly $d_{\text{Hamming}}(\mathcal{P}_1, \mathcal{P}_2) \approx N/2$. It may thus be reasonable to consider only clusters which are several times more homogeneous than the expected null-model homogeneity (in the experiments we have set this threshold to 10 times the null-model homogeneity).

For the clusters which are considered homogeneous, we observe their populations, and out of all models investigated we choose the one leading to the partition cluster of largest size. For this dominating cluster, we then select a prototypical partition. For selecting such prototypical partitions in pairwise clustering problems, we refer the reader to [13], where it is shown that the pairwise clustering problem can be equivalently restated as a $k$-means problem in a suitably chosen embedding space. Each partition is represented as a vector

in this space. This property allows us to select those partitions as representants, which are closest to the partition cluster centroids. The whole work-flow of model selection is summarized schematically in figure 1.

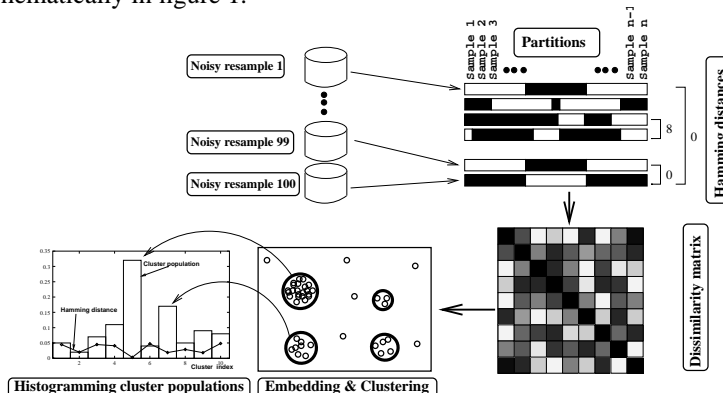

Figure 1: Model selection: schematic work-flow for one fixed value of the $\ell_1$–constraint $\kappa$.

## 4 Experiments

**Clustering USPS digits.** In a first experiment we test our method for the task of clustering digits from the USPS handwritten digits database. Sample images are shown in figure 2.

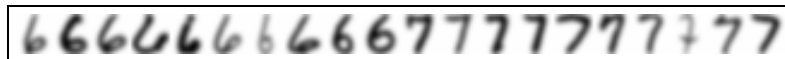

Figure 2: Sample images of digits '6' and '7' from the USPS database.

The $16 \times 16$ gray-value images of the digits are treated as 256-dimensional vectors. For this experiment, we extracted a subset of 200 images, consisting of randomly selected digits '6' and '7'. Based on this dataset, we first selected the most stable model according to the model selection procedure described in section 3. We observed the stability of the solutions for different constraint values $\kappa$ on the interval $[0.7, 1.8]$ with a step-size of $0.1$.

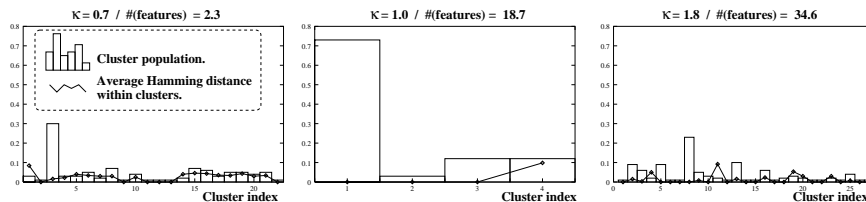

Figure 3: Model selection: three different choices of the $\ell_1$-constraint $\kappa$. The histograms show the relative population of partition clusters. The solid line indicates the average pairwise Hamming distance between partitions (divided by 100).

Figure 3 exemplarily shows the outcomes of the stability analysis: in the left panel, the solution is so highly constrained that on average only 2.3 features (pixels) are selected. One can see that the solutions are rather instable. Subsets of only two features seem to be too small for building a consistent splitting hypothesis. Even the most populated partition cluster (index 3) contains only 30% of all partitions. If, on the other hand, the constraint is relaxed too far, we also arrive at the very instable situation, depicted in the right panel: for $\kappa = 1.8$, on average $34.6$ pixels are selected. Optimizing the model in this 35-dimensional feature space seems to be difficult, probably because the EM-algorithm is often trapped by suboptimal local optima. In-between these models, however, we find a highly stable solution for $\kappa = 1.0$ in moderate dimensions (on average 18.7 features), see the middle panel. In this case, the dominating partition cluster (cluster no. 1 in the histogram) contains almost 75% of all partitions.

Having selected the optimal model parameter $\kappa = 1.0$, in a next step we select the representative partition (= the one nearest to the centroid) of the dominating partition cluster (no. 1 in the middle panel of figure 3). This partition splits the dataset into two clusters, which highly agree with the true labeling. In the upper part of figure 4, both the inferred labels and the true labels are depicted by horizontal bar diagrams. Only three samples out of 200 are mislabeled (the rightmost three samples). The lower panel of this figure shows several rows, each of which represents one automatically selected feature. Each of the 200 grey-value coded pixel blocks in a row indicates the feature value for one sample. For a better visualization, the features (rows) are permuted according to either high values (black) or low values (white) for one of the two clusters.

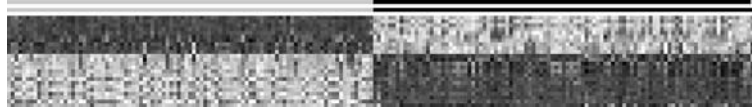

Figure 4: Optimal model: representative partition. Upper horizontal bar: true labels of the 200 samples (black = '6', grey = '7'). Lower bar: inferred labels. Lower panel: each row consists of grey-value coded values of the selected features for all samples (1 pixel block = 1 sample).

We are not only interested in the stability of *splittings of the dataset*, but also in the stability of the *feature selection process*. In order to quantify this latter stability, we return to the dominating partition cluster no. 1 in the middle panel of figure 3, and for each of the 73 partitions in this cluster, we count how often a particular feature has been selected. The 22 features (pixels) which are selected in at least one halve of the partitions, are plotted in the second panel of figure 5. The selection stability is grey-value coded (black = 100% stable). To the left and to the right we have again plotted two typical sample images of both classes from the database. A comparison with the selected features leads us to the conclusion, that we were not only able to find reasonable clusters, but we also have exactly selected those discriminative features which we would have expected in this control experiment. In the rightmost panel, we have also plotted one of the three mislabeled '7's which has been assigned to the '6' cluster.

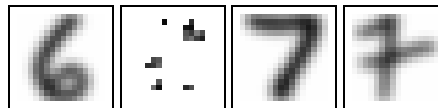

Figure 5: From left to right: **First:** a typical '6'. **Second:** automatically extracted features. **Third:** a typical '7'. **Fourth:** one of the three mislabeled '7's.

**Clustering faces.** In a second experiment we applied our method to the problem of clustering face images. From the *Stirling Faces* database (http://pics.psych.stir.ac.uk/cgi-bin/PICS/New/pics.cgi) we selected all 68 grey-valued front views of faces and all 105 profile views. The images are rather inhomogeneous, since they show different persons with different facial expressions. Some sample images are depicted in figure 6. For a complete overview over the whole image set, we refer the reader to our supplementary web page http://www.cs.uni-bonn.de/∼roth/FACES/split.html, where all images can be viewed in higher quality.

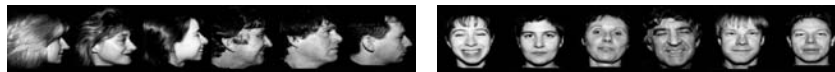

Figure 6: Example images form the *Stirling Faces* database.

Since it appears to be infeasible to work directly on the set of pixels of the high-resolution images, in a first step we extracted the 10 leading *eigenfaces* of the total dataset (eigenfaces are simply the eigenvectors $\boldsymbol{v}_i$ of the images treated as pixel-wise vectorial objects). These eigenfaces are depicted in figure 7. We then applied our method to these image objects, which are represented as 10-dimensional vectors. Note that the original images $I_j$ can be (partially) reconstructed from this truncated eigenvector expansion as $I'_j = \sum_{i=1}^{10} \boldsymbol{v}_i \boldsymbol{v}_i^\top I_j$ (assuming the image vectors $I_j$ to be centered).

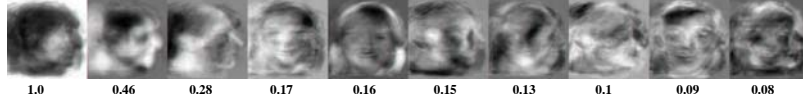

Figure 7: First 10 leading eigenfaces and their relative eigenvalues.

We again start our analysis with selecting an optimal model. Figure 8 depicts the outcome of the model selection procedure. The left panel shows both the number of extracted features and the relative population of the largest partition cluster for different values of $\kappa$. The most stable model is obtained for $\kappa = 1.0$. On average, 3.04 features (eigenfaces) have been selected. A detailed analysis of the selected features within the dominating partition cluster (no. 5 in the right panel) shows that the eigenfaces no. 2, 3 and 7 are all selected with a stability of more than 98%. It is interesting to notice that the leading eigenface no. 1 with the distinctly largest eigenvalue has not been selected.

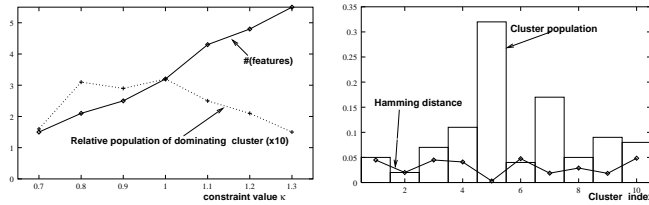

Figure 8: Model selection. Left: average number of selected features and relative population of the dominating partition cluster vs. $\kappa$. Right: partition clusters for optimal model with $\kappa = 1$.

In every M-step of our algorithm, a linear discriminant analysis is performed, in which a weight vector $\beta$ for all features is computed (due to the incorporated feature selection mechanism, most weights will be exactly zero). For a given partition of the objects, the linear combination of the eigenface-features induced by this weight vector is known as the *Fisherface*. Our method can, thus, be interpreted as a clustering method that finds a partition and simultaneously produces a "sparse" Fisherface which consists of a linear combination of the most discriminative eigenfaces. Figure 9 shows the derived Fisherface, reconstructed from the weight vector of the representative partition (no. 5 in the right panel of figure 8). Note that there are only 3 nonzero weights $\beta_2 = 0.8$, $\beta_3 = 0.05$ and $\beta_7 = 0.2$.

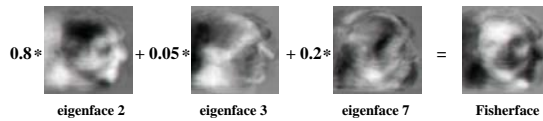

Figure 9: The inferred Fisherface as a linear combination of 3 eigenfaces.

The representative partition of the dominating cluster (no. 5 in the right panel of figure 8) splits the images in two groups, which again highly coincide with the original groups of frontal and profile faces. Only 7 out of all 173 images are mislabeled w.r.t. this "ground-truth" labeling. The success of the clustering method can be understood by reconstructing the original images from the inferred Fisherface (which is nothing but a weighted and truncated eigenvector reconstruction of the original images). Figure 10 shows the same images as in figure 6, this time, however, reconstructed from the Fisherface. For better visualization, all images are rescaled to the full range of 255 grey values. One can see the clear distinction between frontal and profile faces, which mainly results from different signs of the projections of the images on the Fisherface. Again, the whole set of reconstructed images can be viewed on our supplementary material web page in higher quality.

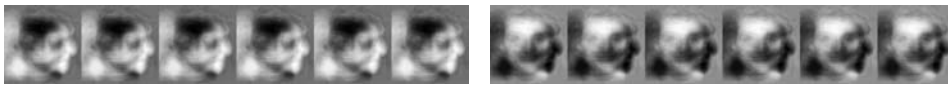

Figure 10: Images from figure 6, reconstructed from the Fisherface.

# 5  Conclusions

The problem tackled in this paper consists of simultaneously clustering objects and automatically extracting subsets of features which are most discriminative for this object partition. Some approaches have been proposed in the literature, most of which, however, bear several inherent shortcomings, such as an unclear probabilistic model, the simplifying assumption of features as being uncorrelated, or the absence of a plausible model selection strategy. The latter issue is of particular importance, since many approaches seem to suffer from ambiguities caused by contradictory splitting hypotheses. In this work we have presented a new approach which has the potential to overcome these shortcomings. It has a clear interpretation in terms of a constrained Gaussian mixture model, which combines a clustering method with a Bayesian inference mechanism for automatically selecting relevant features. We further present an optimization algorithm with guaranteed convergence to a local optimum. The model has only one free parameter, $\kappa$, for which we propose a stability-based model selection procedure. Experiments demonstrate that this method is able to correctly infer partitions and meaningful feature sets.

Our method currently only implements partitions of the object set into two clusters. For finding multiple clusters, we propose to iteratively split the dataset. Such iterative splits have been successfully applied to the problem of simultaneously clustering *gene expression datasets* and selecting relevant genes. Details on these biological applications of our method will appear elsewhere.

**Acknowledgments.** The authors would like to thank Joachim M. Buhmann for helpful discussions and suggestions.

## Footnotes

[1]For an introduction to the ARD principle the reader is referred to [10].

# References

[1] A. Ben-Dor, N. Friedman, and Z. Yakhini. Class discovery in gene expression data. In *Procs. RECOMB*, pages 31–38, 2001.

[2] A.P. Dempster, N.M. Laird, and D.B. Rubin. Maximum likelihood from incomplete data via the EM algorithm. *J. R. Stat. Soc. B*, 39:1–38, 1977.

[3] M. Figueiredo and A. K. Jain. Bayesian learning of sparse classifiers. In *CVPR2001*, pages 35–41, 2001.

[4] T. Hastie, A. Buja, and R. Tibshirani. Penalized discriminant analysis. *Ann. Stat.*, 23:73–102, 1995.

[5] T. Hastie and R. Tibshirani. Discriminant analysis by gaussian mixtures. *J. R. Stat. Soc. B*, 58:158–176, 1996.

[6] T. Hastie, R. Tibshirani, and A. Buja. Flexible discriminant analysis by optimal scoring. *J. Am. Stat. Assoc.*, 89:1255–1270, 1994.

[7] T. Hofmann and J. Buhmann. Pairwise data clustering by deterministic annealing. *IEEE Trans. Pattern Anal. Mach. Intell.*, 19(1):1–14, 1997.

[8] T. Lange, M. Braun, V. Roth, and J.M. Buhmann. Stability-based model selection. In *Advances in Neural Information Processing Systems*, volume 15, 2003. To appear.

[9] M.H. Law, A.K. Jain, and M.A.T. Figueiredo. Feature selection in mixture-based clustering. In *Advances in Neural Information Processing Systems*, volume 15, 2003. To appear.

[10] D.J.C. MacKay. Bayesian non-linear modelling for the prediction competition. In *ASHRAE Transactions Pt.2*, volume 100, pages 1053–1062, Atlanta, Georgia, 1994.

[11] F. Meinecke, A. Ziehe, M. Kawanabe, and K.-R. Müller. Estimating the reliability of ICA projections. In *Advances in Neural Information Processing Systems*, volume 14, 2002.

[12] M. Osborne, B. Presnell, and B. Turlach. On the lasso and its dual. *J. Comput. Graph. Stat.*, 9:319–337, 2000.

[13] V. Roth, J. Laub, J. M. Buhmann, and K.-R. Müller. Going metric: Denoising pairwise data. In *Advances in Neural Information Processing Systems*, volume 15, 2003. To appear.

[14] R.J. Tibshirani. Regression shrinkage and selection via the lasso. *J. R. Stat. Soc. B*, 58(1):267–288, 1996.

[15] A. v.Heydebreck, W. Huber, A. Poustka, and M. Vingron. Identifying splits with clear separation: a new class discovery method for gene expression data. *Bioinformatics*, 17, 2001.
